# Inferring Ground Truth from Subjective Labelling of Venus Images

**Padhraic Smyth, Usama Fayyad**
Jet Propulsion Laboratory 525-3660,
Caltech, 4800 Oak Grove Drive,
Pasadena, CA 91109

**Michael Burl, Pietro Perona**
Department of Electrical Engineering
Caltech, MS 116-81,
Pasadena, CA 91125

**Pierre Baldi***
Jet Propulsion Laboratory 303-310,
Caltech, 4800 Oak Grove Drive,
Pasadena, CA 91109

## Abstract

In remote sensing applications "ground-truth" data is often used as the basis for training pattern recognition algorithms to generate thematic maps or to detect objects of interest. In practical situations, experts may visually examine the images and provide a subjective noisy estimate of the truth. Calibrating the reliability and bias of expert labellers is a non-trivial problem. In this paper we discuss some of our recent work on this topic in the context of detecting small volcanoes in Magellan SAR images of Venus. Empirical results (using the Expectation-Maximization procedure) suggest that accounting for subjective noise can be quite significant in terms of quantifying both human and algorithm detection performance.

## 1 Introduction

In certain pattern recognition applications, particularly in remote-sensing and medical diagnosis, the standard assumption that the labelling of the data has been

carried out in a reasonably objective and reliable manner may not be appropriate. Instead of "ground truth" one may only have the subjective opinion(s) of one or more experts. For example, medical data or image data may be collected off-line and some time later a set of experts analyze the data and produce a set of class labels. The central problem is that of trying to infer the "ground truth" given the noisy subjective estimates of the experts. When one wishes to apply a supervised learning algorithm to the data, the problem is primarily twofold: (i) how to evaluate the relative performance of experts and algorithms, and (ii) how to train a pattern recognition system in the absence of absolute ground truth.

In this paper we focus on problem (i), namely the performance evaluation issue, and in particular we discuss the application of a particular modelling technique to the problem of counting volcanoes on the surface of Venus. For problem (ii), in previous work we have shown that when the inferred labels have a probabilistic interpretation, a simple mixture model argument leads to straightforward modifications of various learning algorithms [1].

It should be noted that the issue of inferring ground truth from subjective labels has appeared in the literature under various guises. French [2] provides a Bayesian perspective on the problem of combining multiple opinions. In the field of medical diagnosis there is a significant body of work on latent variable models for inferring hidden "truth" from subjective diagnoses (e.g., see Uebersax [3]). More abstract theoretical models have also been developed under assumptions of specific labelling patterns (e.g., Lugosi [4] and references therein). The contribution of this paper is twofold: (i) this is the first application of latent-variable subjective-rating models to a large-scale *image* analysis problem as far as we are aware, and (ii) the focus of our work is on the pattern recognition aspect of the problem, i.e., comparing human and algorithmic performance as opposed to simply comparing humans to each other.

## 2 Background: Automated Detection of Volcanoes in Radar Images of Venus

Although modern remote-sensing and sky-telescope technology has made rapid recent advances in terms of data collection capabilities, image analysis often remains a strictly manual process and much investigative work is carried out using hardcopy photographs. The Magellan Venus data set is a typical example: between 1991 and 1994 the Magellan spacecraft transmitted back to earth a data set consisting of over 30,000 high resolution (75m per pixel) synthetic aperture radar (SAR) images of the Venusian surface [5]. This data set is greater than that gathered by all previous planetary missions combined — planetary scientists are literally swamped by data. There are estimated to be on the order of $10^6$ small (less than 15km in diameter) visible volcanoes scattered throughout the 30,000 images [6]. It has been estimated that manually locating all of these volcanoes would require on the order of 10 man-years of a planetary geologist's time to carry out — our experience has been that even a few hours of image analysis severely taxes the concentration abilities of human labellers.

From a scientific viewpoint the ability to accurately locate and characterize the

many volcanoes is a necessary requirement before more advanced planetary geology studies can be carried out: analysis of spatial clustering patterns, correlation with other geologic features, and so forth. From an engineering viewpoint, automation of the volcano detection task presents a significant challenge to current capabilities in computer vision and pattern recognition due to the variability of the volcanoes and the significant background "clutter" present in most of the images. Figure 1 shows a Magellan subimage of size 30km square containing at least 10 small volcanoes.

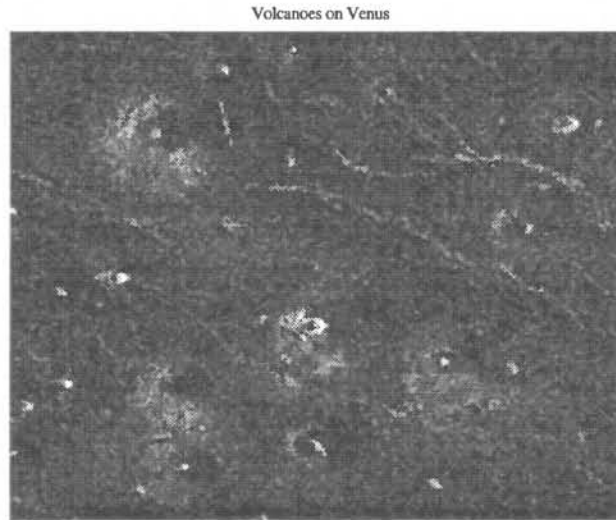

Figure 1: A 30km × 30km region from the Magellan SAR data, which contains a number of small volcanoes.

The purpose of this paper is not to describe pattern recognition methods for volcano detection but rather to discuss some of the issues involved in collecting and calibrating labelled training data from experts. Details of a volcano detection method using matched filtering, SVD projections and a Gaussian classifier are provided in [7].

## 3   Volcano Labelling

Training examples are collected by having the planetary geologists examine an image on the computer screen and then using a mouse to indicate where they think the volcanoes (if any) are located. Typically it can take from 15 minutes to 1 hour to label an image (depending on how many volcanoes are present), where each image represents a 75km square patch on the surface of Venus. An image may contain on the order of 100 volcanoes, although a more typical number is between 30 and 40.

There can be considerable ambiguity in volcano labelling: for the same image, different scientists can produce different label lists, and even the same scientist can produce different lists over time. To address this problem we introduced the notion of having the scientists label training examples into quantized probability

bins or "types", where the probability bins correspond to visually distinguishable sub-categories of volcanoes. In particular, we have used 5 types: (1) summit pits, bright-dark radar pair, and apparent topographic slope, all clearly visible, probability 0.98, (2) only 2 of the 3 criteria of type 1 are visible, probability 0.80, (3) no summit pit visible, evidence of flanks or circular outline, probability 0.60, (4) only a summit pit visible, probability 0.50, (5) no volcano-like features visible, probability 0.0. These subjective probabilities correspond to the mean probability that a volcano exists at a particular location given that it belongs to a particular type and were elicited after considerable discussions with the planetary geologists. Thus, the observed data for each ROI consists of labels $\underline{l}$, which are noisy estimates of true "type" $t$, which in turn is probabilistically related to the hidden event of interest, $v$, the presence of a volcano:

$$p(v|\underline{l}) = \sum_{t=1}^{T} p(v|t)p(t|\underline{l}) \tag{1}$$

where $T$ is the number of types (and labels). The subjective probabilities described above correspond to $p(v|t)$: to be able to infer the probability of a volcano given a set of labels $\underline{l}$ it remains to estimate the $p(t|\underline{l})$ terms.

## 4    Inferring the Label-Type Parameters via the EM Procedure

We follow a general model for subjective labelling originally proposed by Dawid and Skene [8] and apply it to the image labelling problem: more details on this overall approach are provided in [9]. Let $N$ be the number of local regions of interest (ROI's) in the database (these are 15 pixel square image patches for the volcano application). For simplicity we consider the case of just a single labeller who labels a given set of regions of interest (ROIs) a number of times — the extension to multiple labellers is straightforward assuming conditional independence of the labellings given the true type. Let $n_{il}$ be the number of times that ROI $i$ is labelled with label $l$. Let $Y_{it}$ denote a binary variable which takes value 1 if the true type of volcano $i$ is $t^*$, and is 0 otherwise. We assume that labels are assigned independently to a given ROI from one labelling to the next, given that the type is known. If the true type $t^*$ is known then

$$p(\text{observed labels}|t^*, i) \propto \prod_{l=1}^{T} p(l|t)^{n_{il}}. \tag{2}$$

Thus, unconditionally, we have

$$p(\text{observed labels}, t^*|i) \propto \prod_{t=1}^{T} \left( p(t) \prod_{l=1}^{T} p(l|t)^{n_{il}} \right)^{Y_{it}}, \tag{3}$$

where $Y_{it} = 1$ if $t = t^*$ and 0 otherwise. Assuming that each ROI is labelled independently of the others (no spatial correlation in terms of labels),

$$p(\text{observed labels}, t_i^*) \propto \prod_{i}^{N} \prod_{t=1}^{T} \left( p(t) \prod_{l=1}^{T} p(l|t)^{n_{il}} \right)^{Y_{it}}. \tag{4}$$

Still assuming that the types $t$ for each ROI are known (the $Y_{it}$), the maximum likelihood estimators of $p(l|t)$ and $p(t)$ are

$$\hat{p}(l|t) = \frac{\sum_i Y_{it} n_{il}}{\sum_l^T \sum_i Y_{it} n_{il}} \tag{5}$$

and

$$\hat{p}(t) = \frac{1}{N} \sum_i Y_{it}. \tag{6}$$

From Bayes' rule one can then show that

$$p(Y_{it} = 1|\text{observed data}) = \frac{1}{C} \prod_l^T p(l|t)^{n_{il}} p(t) \tag{7}$$

where $C$ is a normalization constant. Thus, given the observed data $n_{il}$ and the parameters $p(l|t)$ and $p(t)$, one can infer the posterior probabilities of each type via Equation 7.

However, without knowing the $Y_{it}$ values we can not infer the parameters $p(l|t)$ and $p(t)$. One can treat the $Y_{it}$ as hidden and thus apply the well-known Expectation-Maximization (EM) procedure to find a local maximum of the likelihood function:

1. Obtain some initial estimates of the expected values of $Y_{it}$, e.g.,

$$E[Y_{it}] = \frac{n_{il}}{\sum_l n_{il}} \tag{8}$$

2. M-step: choose the values of $p(l|t)$ and $p(t)$ which **maximize** the likelihood function (according to Equations 5 and 6), using $E[Y_{it}]$ in place of $Y_{it}$.

3. E-step: calculate the conditional **expectation** of $Y_{it}$, $E[Y_{it}|\text{data}] = p(Y_{it} = 1|\text{data})$ (Equation 7).

4. Repeat Steps 2 and 3 until convergence.

## 5   Experimental Results

### 5.1   Combining Multiple Expert Opinions

Labellings from 4 geologists on the 4 images resulted in 269 possible volcanoes (ROIs) being identified. Application of the EM procedure resulted in label-type probability matrices as shown in Table 1 for Labeller C. The diagonal elements provide an indication of the reliability of the labeller. There is significant miscalibration for label 3's: according to the model, a label 3 from Labeller C is most likely to correspond to type 2. The label-type matrices of all 4 labellers (not shown) indicated that the model placed more weight on the conservative labellers (C and D) than the aggressive ones (A and B).

The determination of posterior probabilities for each of the ROIs is a fundamental step in any quantitative analysis of the volcano data: $p(v|\underline{l}) = \sum_{t=1}^T p(v|t) p(t|\underline{l})$ where the $p(t|\underline{l})$ terms are the posterior probabilities of type given labels provided

Table 1: Type-Label Probabilities for Individual Labellers as estimated via the EM Procedure

Marginal Label Probabilities, Labeller C

| Label 1 | Label 2 | Label 3 | Label 4 | Label 5 |
|---|---|---|---|---|
| 0.026 | 0.056 | 0.193 | 0.416 | 0.309 |

Probability(type|label), Labeller C

|  | Type 1 | Type 2 | Type 3 | Type 4 | Type 5 |
|---|---|---|---|---|---|
| Label 1 | 1.000 | 0.000 | 0.000 | 0.000 | 0.000 |
| Label 2 | 0.019 | 0.977 | 0.004 | 0.000 | 0.000 |
| Label 3 | 0.000 | 0.667 | 0.175 | 0.065 | 0.094 |
| Label 4 | 0.000 | 0.000 | 0.042 | 0.725 | 0.233 |
| Label 5 | 0.000 | 0.000 | 0.389 | 0.000 | 0.611 |

Table 2: 10 ROIs from the database: original scientist labels shown with posterior probabilities estimated via the EM procedure

| ROI | Scientist Labels ($l$) | | | | Posterior Probabilities (EM), $p(t|l)$ | | | | | $p(v|l)$ |
|---|---|---|---|---|---|---|---|---|---|---|
|  | A | B | C | D | Type 1 | Type 2 | Type 3 | Type 4 | Type 5 |  |
| 1 | 4 | 4 | 4 | 5 | 0.000 | 0.000 | 0.000 | 0.816 | 0.184 | 0.408 |
| 2 | 1 | 4 | 4 | 2 | 0.000 | 0.000 | 0.000 | 0.991 | 0.009 | 0.496 |
| 3 | 1 | 1 | 2 | 2 | 0.023 | 0.977 | 0.000 | 0.000 | 0.000 | 0.804 |
| 4 | 3 | 1 | 5 | 3 | 0.000 | 0.000 | 1.000 | 0.000 | 0.000 | 0.600 |
| 5 | 3 | 1 | 3 | 3 | 0.000 | 0.536 | 0.452 | 0.012 | 0.000 | 0.706 |
| 6 | 2 | 2 | 2 | 4 | 0.000 | 1.000 | 0.000 | 0.000 | 0.000 | 0.800 |
| 7 | 3 | 1 | 5 | 5 | 0.000 | 0.000 | 1.000 | 0.000 | 0.000 | 0.600 |
| 8 | 2 | 1 | 4 | 4 | 0.000 | 0.000 | 0.000 | 0.999 | 0.000 | 0.500 |
| 9 | 3 | 2 | 5 | 3 | 0.000 | 0.000 | 0.992 | 0.000 | 0.008 | 0.595 |
| 10 | 4 | 4 | 4 | 4 | 0.000 | 0.000 | 0.000 | 0.996 | 0.004 | 0.498 |

by the EM procedure, and the $p(v|t)$ terms are the subjective volcano-type probabilities discussed in Section 3.2. As shown in Table 2, posterior probabilities for the volcanoes generally are in agreement with intuition and often correspond to taking the majority vote or the "average" of the C and D labels (the conservative labellers). However some $p(v|l)$ estimates could not easily be derived by any simple averaging or voting scheme, e.g., see ROIs 3, 5 and 7 in the table.

## 5.2 Experiment on Comparing Human and Algorithm Performance

The standard receiver operating characteristic (ROC) plots detections versus false alarms [10]. The ROCs shown here differ in two significant ways [11]: (1) the false alarm axis is normalized relative to the number of true positives (necessary since the total number of possible false alarms is not well defined for object detection in images), and (2) the reference labels used in scoring are probabilistic: a detection "scores" $p(v)$ on the detection axis and $1 - p(v)$ on the false alarm axis.

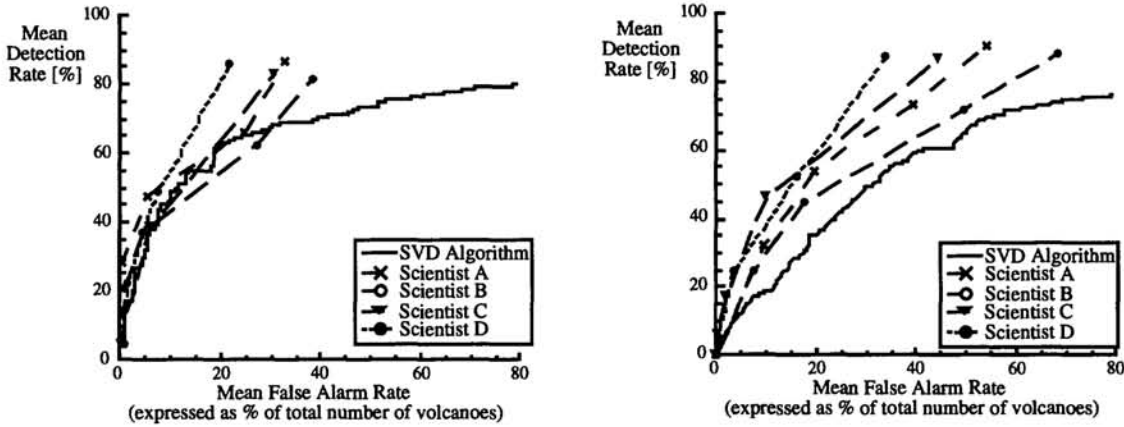

Figure 2: Modified ROCs for both scientists and algorithms: (a) without the labelling or type uncertainty, (b) with full uncertainty model factored in.

As before, data came from 4 images, and there were 269 labelled local regions. The SVD-Gaussian algorithm was evaluated in cross-validation mode (train on 3 images, test on the 4th) and the results combined. The first ROC (Figure 2(a)) does not take into account either label-type or type-volcano probabilities, i.e., the reference list (for algorithm training and overall evaluation) is a consensus list (2 scientists working together) where labels 1,2,3,4 are ignored and all labelled items are counted equally as volcanoes. The individual labellers and algorithm are then scored in the standard "non-weighted" ROC fashion. This curve is optimistic in terms of depicting the accuracy of the detectors since it ignores the underlying probabilistic nature of the labels. Even with this optimistic curve, volcano labelling is relatively inaccurate by either man or machine.

Figure 2(b) shows a weighted ROC: for each of 4 scientists the probabilistic "reference labels" were derived via the EM procedure as in Table 2 from the *other* 3 scientists, and the detections of each scientist were scored according to each such reference set. Performance of the algorithm (the SVD-Gaussian method) was evaluated relative to the EM-derived label estimates of all 4 scientists. Accounting for all of the uncertainty in the data results in a more realistic, if less flattering, set of performance characteristics. The algorithm's performance degrades more than the scientist's performance (for low false alarms rates compared to Figure 2(a)) when the full noise model is used. The algorithm is estimating the posterior probabilities of volcanoes rather poorly and the complete uncertainty model is more sensitive to this fact. This is a function of the SVD feature space rather than the Gaussian classification model.

# 6    Conclusion

Ignoring subjective uncertainty in image labelling can lead to significant over-confidence in terms of performance estimation (for both humans and machines). For the volcano detection task a simple model for uncertainty in the class labels provided insight into the performance of both human and algorithmic detectors. An obvious extension of the maximum likelihood framework outlined here is a Bayesian approach [12]: accounting for parameter uncertainty in the model given the limited amount of training data available is worth investigating.

## Acknowledgements

The research described in this paper was carried out by the Jet Propulsion Laboratory, California Institute of Technology, under a contract with the National Aeronautics and Space Administration and was supported in part by ARPA under grant number N00014-92-J-1860.

## Footnotes

*and Division of Biology, California Institute of Technology

## References

1. P. Smyth, "Learning with probabilistic supervision," in *Computational Learning Theory and Natural Learning Systems 3*, T. Petcshe, M. Kearns, S. Hanson, R. Rivest (eds), Cambridge, MA: MIT Press, to appear.

2. S. French, "Group consensus probability distributions: a critical survey," in *Bayesian Statistics 2*, J. M. Bernardo, M. H. DeGroot, D. V. Lindley, A. F. M. Smith (eds.), Elsevier Science Publishers, North-Holland, pp.183–202, 1985.

3. J. S. Uebersax, "Statistical modeling of expert ratings on medical treatment appropriateness," *J. Amer. Statist. Assoc.*, vol.88, no.422, pp.421–427, 1993.

4. G. Lugosi, "Learning with an unreliable teacher," *Pattern Recognition*, vol. 25, no.1, pp.79–87. 1992.

5. *Science*, special issue on Magellan data, April 12, 1991.

6. J. C. Aubele and E. N. Slyuta, "Small domes on Venus: characteristics and origins," in *Earth, Moon and Planets*, 50/51, 493–532, 1990.

7. M. C. Burl, U. M. Fayyad, P. Perona, P. Smyth, and M. P. Burl, "Automating the hunt for volcanoes on Venus," in *Proceedings of the 1994 Computer Vision and Pattern Recognition Conference: CVPR-94*, Los Alamitos, CA: IEEE Computer Society Press, pp.302–309, 1994.

8. A. P. Dawid and A. M. Skene, "Maximum likelihood estimation of observer error-rates using the EM algorithm," *Applied Statistics*, vol.28, no.1, pp.20–28, 1979.

9. P. Smyth, M. C. Burl, U. M. Fayyad, P. Perona, 'Knowledge discovery in large image databases: dealing with uncertainties in ground truth,' in *Knowledge Discovery in Databases 2*, U. M. Fayyad, G. Piatetsky-Shapiro, P. Smyth, R. Uthurasamy (eds.), AAAI/MIT Press, to appear, 1995.

10. M. S. Chesters, "Human visual perception and ROC methodology in medical imaging," *Phys. Med. Biol.*, vol.37, no.7, pp.1433-1476, 1992.

11. M. C. Burl, U. M. Fayyad, P. Perona, P. Smyth, "Automated analysis of radar imagery of Venus: handling lack of ground truth," in *Proceedings of the IEEE Conference on Image Processing*, Austin, November 1994.

12. W. Buntine, "Operations for learning with graphical models," *Journal of Artificial Intelligence Research*, 2, pp.159–225, 1994.
